# Coding of Naturalistic Stimuli by Auditory Midbrain Neurons

**H. Attias*** and **C.E. Schreiner[†]**
Sloan Center for Theoretical Neurobiology and
W.M. Keck Foundation Center for Integrative Neuroscience
University of California at San Francisco
San Francisco, CA 94143-0444

## Abstract

It is known that humans can make finer discriminations between familiar sounds (e.g. syllables) than between unfamiliar ones (e.g. different noise segments). Here we show that a corresponding enhancement is present in early auditory processing stages. Based on previous work which demonstrated that natural sounds had robust statistical properties that could be quantified, we hypothesize that the auditory system exploits those properties to construct efficient neural codes. To test this hypothesis, we measure the information rate carried by auditory spike trains on narrow-band stimuli whose amplitude modulation has naturalistic characteristics, and compare it to the information rate on stimuli with non-naturalistic modulation. We find that naturalistic inputs significantly enhance the rate of transmitted information, indicating that auditiory neural responses are matched to characteristics of natural auditory scenes.

## 1 Natural Scene Statistics and the Neural Code

A primary goal of hearing research is to understand how complex sounds that occur in natural scenes are processed by the auditory system. However, natural sounds are difficult to describe quantitatively and the complexity of auditory responses they evoke makes it hard to gain insight into their processing. Hence, most studies of auditory physiology are restricted to pure tones and noise stimuli, resulting in a limited understanding of auditory encoding. In this paper we pursue a novel approach to the study of natural sound encoding in auditory spike trains. Our

[†] E-mail: chris@phy.ucsf.edu.

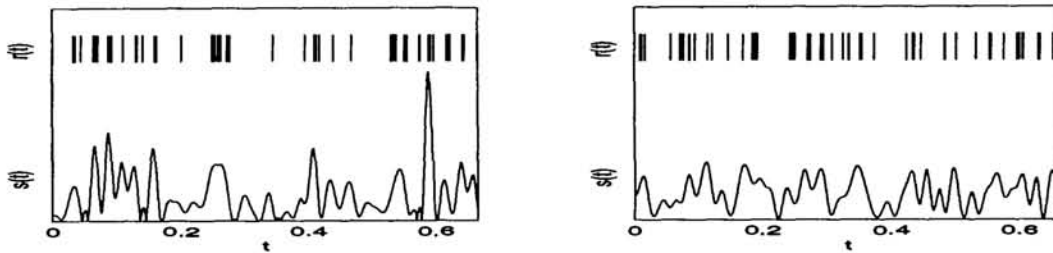

Figure 1: Left: amplitude modulation stimulus drawn from a naturalistic stimulus set, and the evoked spike train of an inferior colliculus neuron. Right: amplitude modulation from a non-naturalistic set and the evoked spike train of the same neuron.

method consists of measuring statistical characteristics of natural auditory scenes, and incorporating them into simple stimuli in a systematic manner, thus creating 'naturalistic' stimuli which enable us to study the encoding of natural sounds in a controlled fashion. The first stage of this program has been described in (Attias and Schreiner 1997); the second is reported below.

Fig. 1 shows two segments of long stimuli and the corresponding spike trains of the same neuron, elicited by pure tones that were amplitude-modulated by these stimuli. While both stimuli appear to be random and to have the same mean and both spike trains have the same firing rate, one may observe that high and low amplitudes are more likely to occur in the stimulus on the left; indeed, these stimuli are drawn from two stimulus sets with different statistical properties. Our present study of auditory coding focuses on assessing the efficiency of this neural code: for a given stimulus set, how well can the animal reconstruct the input sound and discriminate between similar sound segments, based on the evoked spike train, and how those abilities are affected by changing the stimulus statistics. We quantify the discrimination capability of auditory neurons in the inferior colliculus of the cat using concepts from information theory (Bialek et al. 1991; Rieke et al. 1997).

This leads to the issue of optimal coding (Atick 1992). Theoretically, given an auditory scene with particular statistical properties, it is possible to design an encoding scheme that would exploit those properties, resulting in a neural code that is optimal for that scene but is consequently less efficient for other scenes. Here we investigate the hypothesis that the auditory system uses a code that is adapted to natural auditory scenes. This question is addressed by comparing the discrimination capability of auditory neurons between sound segments drawn from a naturalistic stimulus set, to the one for a non-naturalistic set.

## 2   Statistics of Natural Sounds

As a first step in investigating the relation between neural responses and auditory inputs, we studied and quantified temporal statistics of natural auditory scenes (Attias and Schreiner 1997). It is well-known that different locations on the basal membrane respond selectively to different frequency components of the incoming sound $x(t)$ (e.g., Pickles 1988), hence the frequency $\nu$ corresponds to a spatial coordinate, in analogy with retinal location in vision. We therefore analyzed a large database of sounds, including speech, music, animal vocalizations, and background sounds, using various filter banks comprising $0 - 10$kHz. In each frequency band $\nu$, the amplitude $a(t) \geq 0$ and phase $\phi(t)$ of the band-limited signal $x_\nu(t) = a(t)\cos(\nu t + \phi(t))$ were extracted, and the amplitude probability distribution $p(a)$ and auto-correlation

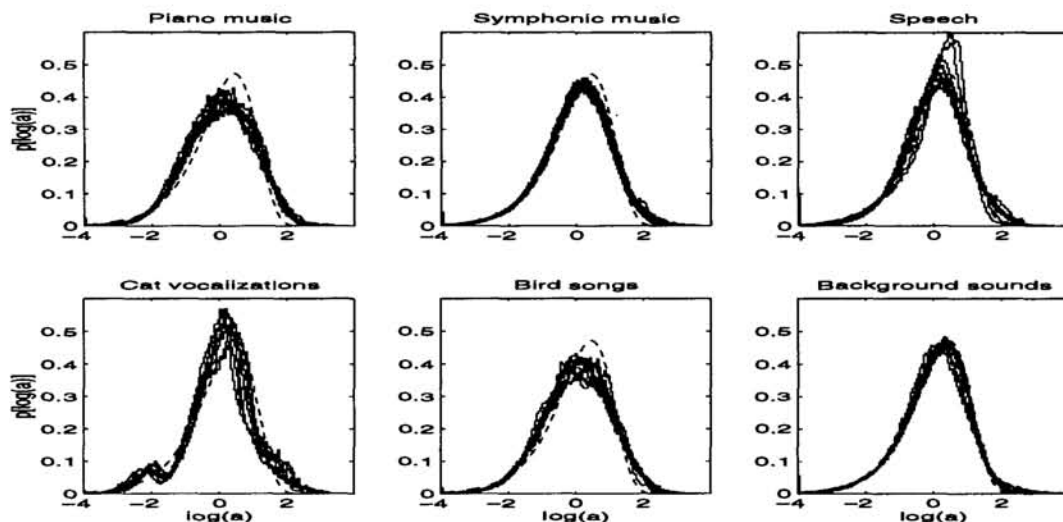

Figure 2: Log-amplitude distribution in several sound ensembles. Different curves for a given ensemble correspond to different frequency bands. The low amplitude peak in the cat plot reflect abundance of silent segments. The theoretical curve $p(\bar{a})$ (1) is plotted for comparison (dashed line).

function $c(\tau) = \langle a(t)a(t+\tau) \rangle$ were computed, as well as those of the instantaneous frequency $d\phi(t)/dt$.

Those statistics were found to be nearly identical in all bands and across all examined sounds. In particular, the distribution of the log-amplitude $\bar{a} = \log a$, normalized to have zero mean and unit variance, could be well-fitted to the form

$$p(\bar{a}) = \beta \exp\left(\beta\bar{a} + \alpha - e^{\beta\bar{a}+\alpha}\right) \tag{1}$$

(with normalization constants $\alpha = -.578$ and $\beta = 1.29$), which should, however, be corrected at large amplitude ($> 5\sigma$). Several examples are displayed in Fig. 1. The log-amplitude distribution (1) corresponds mathematically to the *amplitude* distribution of musical instruments and vocalizations, found to be $p(a) = e^{-a}$ (known as the Laplace distribution in speech signal processing), as well as that of background sounds, where $p(a) \propto ae^{-a^2}$ (which can be shown to be the band amplitude distribution for a Gaussian signal). The power spectra of $a(t)$ (Fourier transform of $c(\tau)$) were found to have a modified $1/f$ form.

Together with the results for $\phi(t)$, those findings show that natural sounds are distinguished from arbitrary ones by robust characteristics. In the present paper we explore to what extent the auditory system exploits them in constructing efficient neural codes. Another important point made by (Attias and Schreiner 1997), as well as by (Ruderman and Bialek 1994) regarding visual signals, is that natural inputs are very often not Gaussian (e.g. (1)), unlike the signals used by conventional system-identification methods often applied to the nervous system. In this paper we use non-Gaussian stimuli to study auditory coding.

## 3  Measuring the Rate of Information Transfer

### 3.1  Experiment

Based on our results for temporal statistics of natural auditory scenes, we can construct 'naturalistic' stimuli by starting with a simple signal and systematically incorporate successively more complicated characteristics of natural sounds into it.

We chose to use narrow-band stimuli consisting of amplitude-modulated carriers $a(t)\cos(\nu t)$ at sound frequencies $\nu = 2 - 9\text{kHz}$ with no phase modulation. Focusing on one-point amplitude statistics, we constructed a white naturalistic amplitude by choosing $a(t)$ from an exponential distribution with a cutoff, $p(0 \leq a \leq a_c) \propto e^{-a}$, $p(a > a_c) = 0$ at each time point $t$ independently, using a cutoff modulation frequency of $f_c = 100\text{Hz}$ (i.e., $| \tilde{a}(f \leq f_c) |= const.$, $| \tilde{a}(f > f_c) |= 0$, where $\tilde{a}(f)$ is the Fourier transform of $a(t)$). We also used a non-naturalistic stimulus set where $a(t)$ was chosen from a uniform distribution $p(0 \leq a \leq b_c) = 1/b_c$, $p(a > b_c) = 0$, with $b_c$ adjusted so that both stimulus sets had the same mean. A short segment from each set is shown in Fig. 1, and the two distributions are plotted in Figs. 3,4 (right).

Stimuli of $15 - 20\text{min}$ duration were played to ketamine-anesthetized cats. To minimize adaptation effects we alternated between the two sets using 10sec long segments. Single-unit recordings were made from the inferior colliculus (IC), a subthalamic auditory processing stage (e.g., Pickles 1988). Each IC unit responds best to a narrow range of sound frequencies, the center of which is called its 'best frequency' (BF). Neighboring units have similar BF's, in accord with the topographic frequency organization of the auditory system. For each unit, stimuli with carrier frequency $\nu$ at most 500Hz away from the unit's BF were used. Firing rates in response to those stimuli were between $60 - 100\text{Hz}$. The stimulus and the electrode signal were recorded simultaeneously at a sampling rate of 24kHz. After detecting and sotring the spikes and extracting the stimulus amplitude, both amplitude and spike train were down-sampled to 3kHz.

## 3.2 Analysis

In order to assess the ability to discriminate between different inputs based on the observed spike train, we computed the mutual information $I_{r,s}$ between the spike train response $r(t) = \sum_i \delta(t - t_i)$, where $t_i$ are the spike times, and the stimulus amplitude $s(t)$. I consists of two terms, $I_{r,s} = H_s - H_{s|r}$, where $H_s$ is the stimulus entropy (the log-number of different stimuli) and $H_{s|r}$ is the entropy of the stimulus conditioned on the response (the log-number of different stimuli that could elicit a given response, and thus could not be discriminated based on that response, averaged over all responses). Our approach generally follows the ideas of (Bialek et al. 1991; Rieke et al. 1997).

To simplify the calculation, we first modified the stimuli $s(t)$ to get $s'(t) = f(s(t))$, where the function $f(s)$ was chosen so that $s'$ was Gaussian. Hence for exponential stimuli $f(s) = \sqrt{(2)}\text{erfi}(1 - 2e^{-s})$ and for uniform stimuli $f(s) = \sqrt{(2)}\text{erfi}(2s/b_c - 1)$, where erfi is the inverse error function. This Gaussianization has two advantages: first, the expression for the mutual information $I_{r,s'}$ ($= I_{r,s}$) is now simpler, being given by the frequency-dependent signal-to-noise ratio $\text{SNR}(f)$ (see below), since $H_{s'}$ depends only on the power spectrum of $s'(t)$; second and more importantly, the noise distribution was observed to become closer to Gaussian following this transformation.

To compute $H_{s'|r}$ we bound it from above by $\int_0^{f_c} df H[\tilde{s}'(f) | \tilde{r}(f)]$, the calculation of which requires the conditional distribution $p[\tilde{s}'(f) | \tilde{r}(f)]$ (note that these variables are complex, hence this is the joint ditribution of the real and imaginary parts). The latter is approximated by a Gaussian with mean $\tilde{s}'_r(f)$ and variance $N_r(f)$. This variance is, in fact, the power spectrum of the noise, $N_r(f) = \langle | \tilde{n}_r(f) |^2 \rangle$, which we define by $n_r(t) = s'(t) - s'_r(t)$. Computing the mutual information for those Gaussian distributions is straightforward and provides a lower bound on the

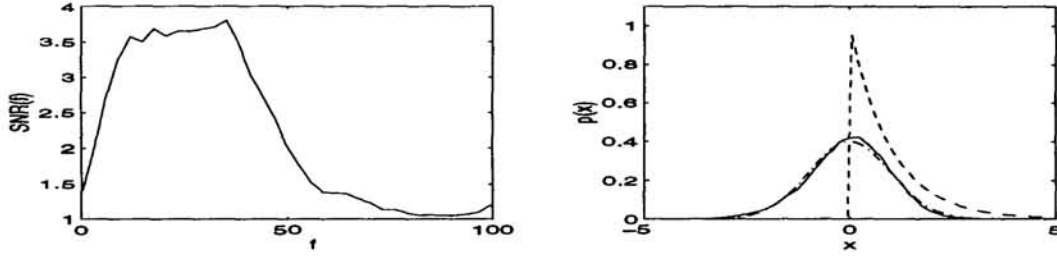

Figure 3: Left: signal-to-noise ratio SNR($f$) vs. modulation frequency $f$ for naturalistic stimuli. Right: normalized noise distribution (solid line), amplitude distribution of stimuli (dashed line) and of Gaussianized stimuli (dashed-dotted line).

true $I_{r,s}$,

$$I_{r,s} = I_{r,s'} \geq \int_0^{f_c} df \log_2 \mathrm{SNR}(f) \ . \tag{2}$$

The signal-to-noise ratio is given by $\mathrm{SNR}(f) = S'(f)/\langle N_r(f)\rangle_r$, where $S'(f) = \langle| \tilde{s}'(f) |^2 \rangle$ is the spectrum of the Gaussianized stimulus and the averaging $\langle \cdot \rangle_r$ is performed over all responses.

The main object here is $\tilde{s}'_r(f)$, which is an estimate of the stimulus from the elicited spike train, and would optimally be given by the conditional mean $\int d\tilde{s}' \tilde{s}' p(\tilde{s}' \mid \tilde{r})$ at each $f$ (Kay 1993). For Gaussian $p(\tilde{s}', \tilde{r})$ this estimator, which is generally non-linear, becomes linear in $\tilde{r}(f)$ and is given by $\tilde{h}(f)\tilde{r}(f)$, where $\tilde{h}(f) = \langle \tilde{s}'(f)\tilde{r}^\star(f)\rangle/\langle \tilde{r}(f)\tilde{r}^\star(f)\rangle$ is the Wiener filter. However, since our distributions were only approximately Gaussians we used the conditional mean, obtained by the kernel estimate

$$\tilde{s}'_r(f) \;=\; \sqrt{S'(f)}\sum_i \int df' \frac{\tilde{s}'_i(f')}{\sqrt{S'(f')}} k_i(f,f') / \sum_j \int df'' k_j(f,f'') \ ,$$

$$k_i(f,f') \;=\; k\left[ \frac{\tilde{r}(f)}{\sqrt{R(f)}} - \frac{\tilde{r}_i(f')}{\sqrt{R(f')}} \right] \tag{3}$$

where $k$ is a Gaussian kernel, $R(f)$ is the spectrum of the spike train, and $i$ indexes the data points obtained by computing FFT using a sliding window. The scaling by $\sqrt{S'}, \sqrt{R}$ reflects the assumption that the distributions at all $f$ differ only by their variance, which enables us to use the data points at all frequencies to estimate $\tilde{s}'$ at a given $f$. Our estimate produced a slightly higher SNR($f$) than the Wiener estimate used by (Bialek et al. 1991;Rieke et al. 1997) and others.

## 4   Information on Naturalistic Stimuli

The SNR($f$) for exponential stimuli is shown in Fig. 3 (left) for one of our units. IC neurons have a preferred modulation frequency $f_m$ (e.g., Pickles 1988), which is about 40Hz for this unit; notice that generally SNR($f$) $\geq 1$, with equality when the stimulus and response are completely independent. Thus, stimulus components at frequencies higher than 60Hz effectively cannot be estimated from the spike train. The stimulus amplitude distribution is shown in Fig. 3 (right, dashed line), together with the noise distribution (normalized to have unit variance; solid line) which is nearly Gaussian.

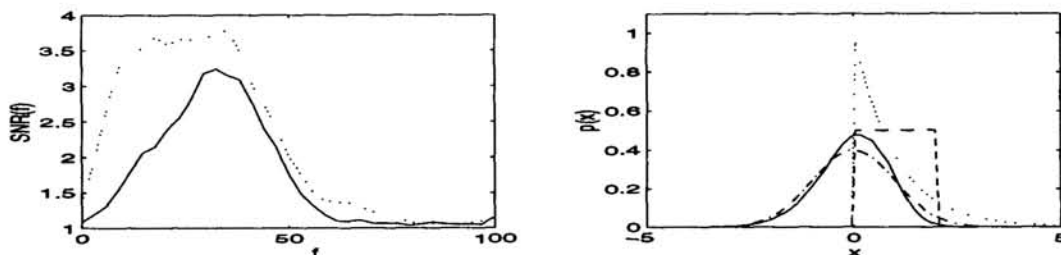

Figure 4: Left: signal-to-noise ratio SNR($f$) vs. modulation frequency $f$ for non-naturalistic stimuli (solid line) compared with naturalistic stimuli (dotted line). Right: normalized noise distribution (solid line), amplitude distribution of stimuli (dashed line) compared with that of naturalistic stimuli (dotted line), and of Gaussianized stimuli (dashed-dotted line).

Using (2) we obtain an information rate of $I_{r,s} \geq 114$bit/sec. For the spike rate of 82spike/sec measured in this unit, this translates into 1.4bit/spike. Averaging across units, we have $1.3 \pm 0.2$bit/spike for naturalistic stimuli.

Although this information rate was computed using the conditional mean estimator (3), it is interesting to examine the Wiener filter $h(t)$ which provides the optimal linear estimator of the stimulus, as discussed in the previous section. This filter is displayed in Fig. 5 (solid line) and has a temporal width of several tens of milliseconds.

## 5    Information on Non-Naturalistic Stimuli

The SNR($f$) for uniform stimuli is shown in Fig. 4 (left, solid line) for the same unit as in Fig. 3, and is significantly lower than the corresponding SNR($f$) for exponential stimuli plotted for comparison (dashed line). For the mutual information rate we obtain $I_{r,s} \geq 77$bit/sec, which amounts to 0.94bit/spike. Averaging across units, we have $0.8 \pm 0.2$bit/spike for non-naturalistic stimuli.

The stimulus amplitude distribution is shown in Fig. 4 (right, dashed line), together with the exponential distribution (dotted line) plotted for comparison, as well as the noise distribution (normalized to have unit variance). The noise in this case is less Gaussian than for exponential stimuli, suggesting that our calculated bound on $I_{r,s}$ may be lower for uniform stimuli.

Fig. 5 shows the stimulus reconstruction filter (dashed line). It has a similar time course as the filter for exponential stimuli, but the decay is significantly slower and its temporal width is more than 100msec.

## 6    Conclusion

We measured the rate at which auditory neurons carry information on simple stimuli with naturalistic amplitude modulation, and found that it was higher than for stimuli with non-naturalistic modulation. A result along the same lines for the frog was obtained by (Rieke et al. 1995) using Gaussian signals whose spectrum was shaped according to the frog call spectrum. Similarly, work in vision (Laughlin 1981; Field 1987; Atick and Redlich 1990; Ruderman and Bialek 1994; Dong and Atick 1995) suggests that visual receptive field properties are consistent with optimal coding predictions based on characteristics of natural images. Future work will explore coding of stimuli with more complex natural statistical characteristics and

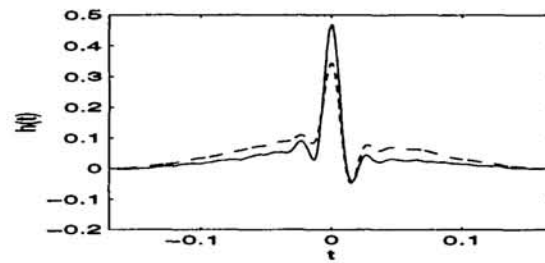

Figure 5: Impulse response of Wiener reconstruction filter for naturalistic stimuli (solid line) and non-naturalistic stimuli (dashed line).

will extend to higher processing stages.

## Acknowledgements

We thank W. Bialek, K. Miller, S. Nagarajan, and F. Theunissen for useful discussions and B. Bonham, M. Escabi, M. Kvale, L. Miller, and H. Read for experimental support. Supported by The Office of Naval Research (N00014-94-1-0547), NIDCD (R01-02260), and the Sloan Foundation.

## Footnotes

\* Corresponding author. E-mail: hagai@phy.ucsf.edu.

## References

J.J. Atick and N. Redlich (1990). Towards a theory of early visual processing. Neural Comput. **2**, 308-320.

J.J. Atick (1992). Could information theory provide an ecological theory of sensory processing. Network **3**, 213-251.

H. Attias and C.E. Schreiner (1997). Temporal low-order statistics of natural sounds. In *Advances in Neural Information Processing Systems 9*, MIT Press.

W. Bialek, F. Rieke, R. de Ruyter van Steveninck, and D. Warland (1991). Reading the neural code. *Science* **252**, 1854-1857.

D.W. Dong and J.J. Atick (1995). Temporal decorrelation: a theory of lagged and non-lagged responses in the lateral geniculate nucleus. Network **6**, 159-178.

D.J. Field (1987). Relations between the statistics of natural images and the response properties of cortical cells. J. Opt. Soc. Am. **4**, 2379-2394.

S.M. Kay (1993). *Fundamentals of Statistical Signal Processing: Estimation Theory.* Prentice-Hall, New Jersey.

S.B. Laughlin (1981). A simple coding procedure enhances a neuron's information capacity. Z. Naturforsch. **36c**, 910-912.

J.O. Pickles (1988). *An introduction to the physiology of hearing* (2nd Ed.). San Diego, CA: Academic Press.

F. Rieke, D. Bodnar, and W. Bialek (1995). Naturalistic stimuli increase the rate and efficiency of information transmission by primary auditory neurons. *Proc. R. Soc. Lond. B*, **262**, 259-265.

F. Rieke, D. Warland, R. de Ruyter van Steveninck, and W. Bialek (1997). *Spikes: Exploring the Neural Code.* MIT Press, Cambridge, MA.

D.L. Ruderman and W. Bialek (1994). Statistics of natural images: scaling in the woods. Phys. Rev. Lett. **73**, 814-817.